# In-Network PCA and Anomaly Detection

**Ling Huang**
University of California
Berkeley, CA 94720
hling@cs.berkeley.edu

**XuanLong Nguyen**
University of California
Berkeley, CA 94720
xuanlong@cs.berkeley.edu

**Minos Garofalakis**
Intel Research
Berkeley, CA 94704
minos.garofalakis@intel.com

**Michael I. Jordan**
University of California
Berkeley, CA 94720
jordan@cs.berkeley.edu

**Anthony Joseph**
University of California
Berkeley, CA 94720
adj@cs.berkeley.edu

**Nina Taft**
Intel Research
Berkeley, CA 94704
nina.taft@intel.com

## Abstract

We consider the problem of network anomaly detection in large distributed systems. In this setting, Principal Component Analysis (PCA) has been proposed as a method for discovering anomalies by continuously tracking the projection of the data onto a residual subspace. This method was shown to work well empirically in highly aggregated networks, that is, those with a limited number of large nodes and at coarse time scales. This approach, however, has scalability limitations. To overcome these limitations, we develop a PCA-based anomaly detector in which adaptive local data filters send to a coordinator just enough data to enable accurate global detection. Our method is based on a stochastic matrix perturbation analysis that characterizes the tradeoff between the accuracy of anomaly detection and the amount of data communicated over the network.

## 1 Introduction

The area of distributed computing systems provides a promising domain for applications of machine learning methods. One of the most interesting aspects of such applications is that learning algorithms that are embedded in a distributed computing infrastructure are themselves part of that infrastructure and must respect its inherent local computing constraints (e.g., constraints on bandwidth, latency, reliability, etc.), while attempting to aggregate information across the infrastructure so as to improve system performance (or availability) in a global sense.

Consider, for example, the problem of detecting anomalies in a wide-area network. While it is straightforward to embed learning algorithms at local nodes to attempt to detect node-level anomalies, these anomalies may not be indicative of network-level problems. Indeed, in recent work, [8] demonstrated a useful role for Principal Component Analysis (PCA) to detect network anomalies. They showed that the minor components of PCA (the subspace obtained after removing the components with largest eigenvalues) revealed anomalies that were not detectable in any single node-level trace. This work assumed an environment in which all the data is continuously pushed to a central site for off-line analysis. Such a solution cannot scale either for networks with a large number of monitors nor for networks seeking to track and detect anomalies at very small time scales.

Designing scalable solutions presents several challenges. Viable solutions need to process data "in-network" to intelligently control the frequency and size of data communications. The key underlying problem is that of developing a mathematical understanding of how to trade off quantization arising from local data filtering against fidelity of the detection analysis. We also need to understand how this tradeoff impacts overall detection accuracy. Finally, the implementation needs to be simple if it is to have impact on developers.

In this paper, we present a simple algorithmic framework for network-wide anomaly detection that relies on distributed tracking combined with approximate PCA analysis, together with supporting theoretical analysis. In brief, the architecture involves a set of local monitors that maintain parameterized sliding filters. These sliding filters yield quantized data streams that are sent to a coordinator. The coordinator makes global decisions based on these quantized data streams. We use stochastic matrix perturbation theory to both assess the impact of quantization on the accuracy of anomaly detection, and to design a method that selects filter parameters in a way that bounds the detection error. The combination of our theoretical tools and local filtering strategies results in an in-network tracking algorithm that can achieve high detection accuracy with low communication overhead; for instance, our experiments show that, by choosing a relative eigen-error of $1.5\%$ (yielding, approximately, a 4% missed detection rate and a 6% false alarm rate), we can filter out more than $90\%$ of the traffic from the original signal.

**Prior Work.** The original work on a PCA-based method by Lakhina et al. [8] has been extended by [17], who show how to infer network anomalies in both spatial and temporal domains. As with [8], this work is completely centralized. [14] and [1] propose distributed PCA algorithms distributed across blocks of rows or columns of the data matrix; however, these methods are not applicable to our case. Furthermore, neither [14] nor [1] address the issue of continuously tracking principal components within a given error tolerance or the issue of implementing a communication/accuracy tradeoff; issues which are the main focus of our work. Other initiatives in distributed monitoring, profiling and anomaly detection aim to share information and foster collaboration between widely distributed monitoring boxes to offer improvements over isolated systems [12, 16]. Work in [2, 10] posits the need for scalable detection of network attacks and intrusions. In the setting of simpler statistics such as sums and counts, in-network detection methods related to ours have been explored by [6]. Finally, recent work in the machine learning literature considers distributed constraints in learning algorithms such as kernel-based classification [11] and graphical model inference [7]. (See [13] for a survey).

## 2 Problem description and background

We consider a monitoring system comprising a set of *local monitor nodes* $M_1, \ldots, M_n$, each of which collects a locally-observed time-series data stream (Fig. 1(a)). For instance, the monitors may collect information on the number of TCP connection requests per second, the number of DNS transactions per minute, or the volume of traffic at port 80 per second. A central *coordinator node* aims to continuously monitor the global collection of time series, and make global decisions such as those concerning matters of network-wide health. Although our methodology is generally applicable, in this paper we focus on the particular application of detecting *volume anomalies*. A volume anomaly refers to unusual traffic load levels in a network that are caused by anomalies such as worms, distributed denial of service attacks, device failures, misconfigurations, and so on.

Each monitor collects a new data point at every time step and, assuming a naive, "continuous push" protocol, sends the new point to the coordinator. Based on these updates, the coordinator keeps track of a sliding time window of size $m$ (i.e., the $m$ most recent data points) for each monitor time series, organized into a matrix $\mathbf{Y}$ of size $m \times n$ (where the $i^{th}$ column $\mathbf{Y}_i$ captures the data from monitor $i$, see Fig. 1(a)). The coordinator then makes its decisions based solely on this (global) $\mathbf{Y}$ matrix.

In the network-wide volume anomaly detection algorithm of [8] the local monitors measure the total volume of traffic (in bytes) on each network link, and periodically (e.g., every 5 minutes) centralize the data by pushing all recent measurements to the coordinator. The coordinator then performs PCA on the assembled $\mathbf{Y}$ matrix to detect volume anomalies. This method has been shown to work remarkably well, presumably due to the inherently low-dimensional nature of the underlying data [9]. However, such a "periodic push" approach suffers from inherent limitations: To ensure fast detection, the update periods should be relatively small; unfortunately, small periods also imply increased monitoring communication overheads, which may very well be unnecessary (e.g., if there are no significant local changes across periods). Instead, in our work, we study how the monitors can effectively filter their time-series updates, sending as little data as possible, yet enough so as to allow the coordinator to make global decisions accurately. We provide analytical bounds on the errors that occur because decisions are made with incomplete data, and explore the tradeoff between reducing data transmissions (communication overhead) and decision accuracy.

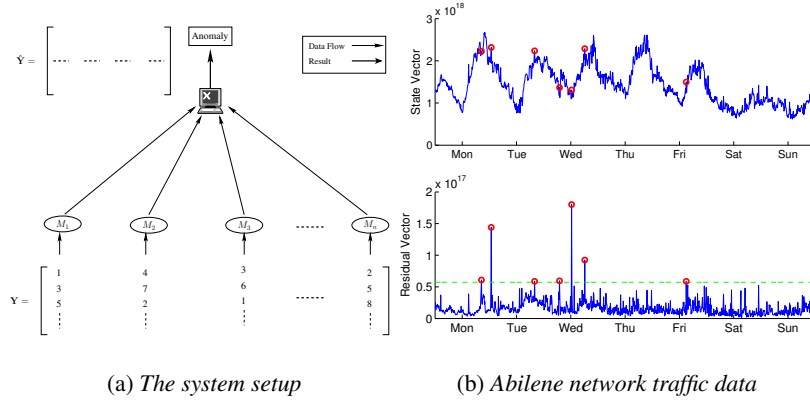

(a) *The system setup*        (b) *Abilene network traffic data*

Figure 1: (a) The distributed monitoring system; (b) Data sample ($\|\mathbf{y}\|^2$) collected over one week (top); its projection in residual subspace (bottom). Dashed line represents a threshold for anomaly detection.

**Using PCA for centralized volume anomaly detection.** As observed by Lakhina et al. [8], due to the high level of traffic aggregation on ISP backbone links, volume anomalies can often go unnoticed by being "buried" within normal traffic patterns (e.g., the circle dots shown in the top plot in Fig 1(b)). On the other hand, they observe that, although, the measured data is of seemingly high dimensionality ($n$ = number of links), normal traffic patterns actually lie in a very low-dimensional subspace; furthermore, separating out this normal traffic subspace using PCA (to find the principal traffic components) makes it much easier to identify volume anomalies in the remaining subspace (bottom plot of Fig. 1(b)).

As before, let $\mathbf{Y}$ be the global $m \times n$ time-series data matrix, centered to have zero mean, and let $\mathbf{y} = \mathbf{y}(t)$ denote a $n$-dimensional vector of measurements (for all links) from a single time step $t$. Formally, PCA is a projection method that maps a given set of data points onto principal components ordered by the amount of data variance that they capture. The set of $n$ principal components, $\{\mathbf{v}_i\}_{i=1}^n$, are defined as:

$$\mathbf{v}_i = \arg \max_{\|\mathbf{x}\|=1} \|(\mathbf{Y} - \sum_{j=1}^{i-1} \mathbf{Y}\mathbf{v}_j\mathbf{v}_j^T)\mathbf{x}\|$$

and are the $n$ eigenvectors of the estimated covariance matrix $\mathbf{A} := \frac{1}{m}\mathbf{Y}^T\mathbf{Y}$. As shown in [9], PCA reveals that the Origin-Destination (OD) flow matrices of ISP backbones have low intrinsic dimensionality: For the Abilene network with $41$ links, most data variance can be captured by the first $k = 4$ principal components. Thus, the underlying normal OD flows effectively reside in a (low) $k$-dimensional subspace of $\mathbb{R}^n$. This subspace is referred to as the *normal* traffic subspace $\mathcal{S}_{no}$. The remaining $(n - k)$ principal components constitute the *abnormal* traffic subspace $\mathcal{S}_{ab}$.

Detecting volume anomalies relies on the decomposition of link traffic $\mathbf{y} = \mathbf{y}(t)$ at any time step into normal and abnormal components, $\mathbf{y} = \mathbf{y}_{no} + \mathbf{y}_{ab}$, such that (a) $\mathbf{y}_{no}$ corresponds to modeled normal traffic (the projection of $\mathbf{y}$ onto $\mathcal{S}_{no}$), and (b) $\mathbf{y}_{ab}$ corresponds to residual traffic (the projection of $\mathbf{y}$ onto $\mathcal{S}_{ab}$). Mathematically, $\mathbf{y}_{no}(t)$ and $\mathbf{y}_{ab}(t)$ can be computed as

$$\mathbf{y}_{no}(t) = \mathbf{P}\mathbf{P}^T\mathbf{y}(t) = \mathbf{C}_{no}\mathbf{y}(t) \quad \text{and} \quad \mathbf{y}_{ab}(t) = (\mathbf{I} - \mathbf{P}\mathbf{P}^T)\mathbf{y}(t) = \mathbf{C}_{ab}\mathbf{y}(t)$$

where $\mathbf{P} = [\mathbf{v_1}, \mathbf{v_2}, \ldots, \mathbf{v_k}]$ is formed by the first $k$ principal components which capture the dominant variance in the data. The matrix $\mathbf{C}_{no} = \mathbf{P}\mathbf{P}^T$ represents the linear operator that performs projection onto the normal subspace $\mathcal{S}_{no}$, and $\mathbf{C}_{ab}$ projects onto the abnormal subspace $\mathcal{S}_{ab}$.

As observed in [8], a volume anomaly typically results in a large change to $\mathbf{y}_{ab}$; thus, a useful metric for detecting abnormal traffic patterns is the squared prediction error (SPE):

$$\mathbf{SPE} \equiv \|\mathbf{y}_{ab}\|^2 = \|\mathbf{C}_{ab}\mathbf{y}\|^2$$

(essentially, a quadratic residual function). More formally, their proposed algorithm signals a volume anomaly if $\mathbf{SPE} > Q_\alpha$, where $Q_\alpha$ denotes the threshold statistic for the $\mathbf{SPE}$ residual function at the $1 - \alpha$ confidence level. Such a statistical test for the $\mathbf{SPE}$ residual function, known as the $Q$-statistic [4], can be computed as a function $Q_\alpha = Q_\alpha(\lambda_{k+1}, \ldots, \lambda_n)$ of the $(n-k)$ non-principal eigenvalues of the covariance matrix $\mathbf{A}$.

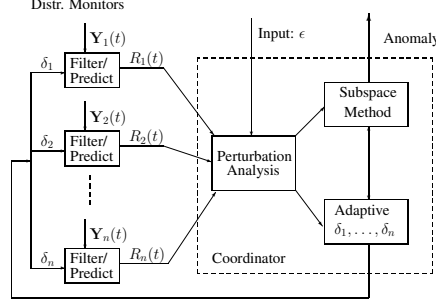

Figure 2: Our in-network tracking and detection framework.

# 3 In-network PCA for anomaly detection

We now describe our version of an anomaly detector that uses distributed tracking and approximate PCA analysis. A key idea is to curtail the amount of data each monitor sends to the coordinator. Because our job is to catch anomalies, rather than to track ongoing state, we point out that the coordinator only needs to have a good approximation of the state when an anomaly is near. It need not track global state very precisely when conditions are normal. This observation makes it intuitive that a reduction in data sharing between monitors and the coordinator should be possible. We curtail the amount of data flow from monitors to the coordinator by installing *local filters* at each monitor. These filters maintain a *local constraint*, and a monitor only sends the coordinator an update of its data when the constraint is violated. The coordinator thus receives an approximate, or "perturbed," view of the data stream at each monitor and hence of the global state. We use stochastic matrix perturbation theory to analyze the effect on our PCA-based anomaly detector of using a perturbed global matrix. Based on this, we can choose the filtering parameters (i.e., the local constraints) so as to limit the effect of the perturbation on the PCA analysis and on any deterioration in the anomaly detector's performance. All of these ideas are combined into a simple, adaptive distributed protocol.

## 3.1 Overview of our approach

Fig. 2 illustrates the overall architecture of our system. We now describe the functionality at the monitors and the coordinator. The goal of a monitor is to track its local raw time-series data, and to decide when the coordinator needs an update. Intuitively, if the time series does not change much, or doesn't change in a way that affects the global condition being tracked, then the monitor does not send anything to the coordinator. The coordinator assumes that the most recently received update is still approximately valid. The update message can be either the current value of the time series, or a summary of the most recent values, or any function of the time series. The update serves as a *prediction* of the future data, because should the monitor send nothing in subsequent time intervals, then the coordinator uses the most recently received update to predict the missing values.

For our anomaly detection application, we filter as follows. Each monitor $i$ maintains a filtering window $F_i(t)$ of size $2\delta_i$ centered at a value $R_i$ (i.e., $F_i(t) = [R_i(t) - \delta_i, R_i(t) + \delta_i]$). At each time $t$, the monitor sends both $\mathbf{Y}_i(t)$ and $R_i(t)$ to the coordinator only if $\mathbf{Y}_i(t) \notin F_i$, otherwise it sends nothing. The window parameter $\delta_i$ is called the *slack*; it captures the amount the time series can drift before an update to the coordinator needs to be sent. The center parameter $R_i(t)$ denotes the approximate representation, or summary, of $\mathbf{Y}_i(t)$. In our implementation, we set $R_i(t)$ equal to the average of last five signal values observed locally at monitor $i$. Let $t^*$ denote the time of the most recent update happens. The monitor needs to send both $\mathbf{Y}_i(t^*)$ and $R_i(t^*)$ to the coordinator when it does an update, because the coordinator will use $\mathbf{Y}_i(t^*)$ at time $t^*$ and $R_i(t^*)$ for all $t > t^*$ until the next update arrives. For any subsequent $t > t^*$ when the coordinator receives no update from that monitor, it will use $R_i(t^*)$ as the prediction for $\mathbf{Y}_i(t)$.

The role of the coordinator is twofold. First, it makes global anomaly-detection decisions based upon the received updates from the monitors. Secondly, it computes the filtering parameters (i.e., the slacks $\delta_i$) for all the monitors based on its view of the global state and the condition for triggering an anomaly. It gives the monitors their slacks initially and updates the value of their slack parameters when needed. Our protocol is thus adaptive. Due to lack of space we do not discuss here the method for deciding when slack updates are needed. The global detection task is the same as in the

centralized scheme. In contrast to the centralized setting, however, the coordinator does not have an exact version of the raw data matrix $\mathbf{Y}$; it has the approximation $\hat{\mathbf{Y}}$ instead. The PCA analysis, including the computation of $\mathcal{S}_{ab}$ is done on the *perturbed* covariance matrix $\hat{\mathbf{A}} := \mathbf{A} - \mathbf{\Delta}$. The magnitude of the perturbation matrix $\mathbf{\Delta}$ is determined by the slack variables $\delta_i$ ($i = 1, \ldots, M$).

## 3.2 Selection of filtering parameters

A key ingredient of our framework is a practical method for choosing the slack parameters $\delta_i$. This choice is critical because these parameters balance the tradeoff between the savings in data communication and the loss of detection accuracy. Clearly, the larger the slack, the less the monitor needs to send, thus leading to both more reduction in communication overhead and potentially more information loss at the coordinator. We employ *stochastic matrix perturbation theory* to quantify the effects of the perturbation of a matrix on key quantities such as eigenvalues and the eigen-subspaces, which in turn affect the detection accuracy.

Our approach is as follows. We measure the size of a perturbation using a norm on $\mathbf{\Delta}$. We derive an upper bound on the changes to the eigenvalues $\lambda_i$ and the residual subspace $\mathbf{C}_{ab}$ as a function of $\|\mathbf{\Delta}\|$. We choose $\delta_i$ to ensure that an approximation to this upper bound on $\mathbf{\Delta}$ is not exceeded. This in turn ensures that $\lambda_i$ and $\mathbf{C}_{ab}$ do not exceed their upper bounds. Controlling these latter terms, we are able to bound the false alarm probability.

Recall that the coordinator's view of the global data matrix is the perturbed matrix $\hat{\mathbf{Y}} = \mathbf{Y} + \mathbf{W}$, where all elements of the column vector $\mathbf{W}_i$ are bounded within the interval $[-\delta_i, \delta_i]$. Let $\lambda_i$ and $\hat{\lambda}_i$ ($i = 1, \ldots, n$) denote the eigenvalues of the covariance matrix $\mathbf{A} = \frac{1}{m}\mathbf{Y}^T\mathbf{Y}$ and its perturbed version $\hat{\mathbf{A}} := \frac{1}{m}\hat{\mathbf{Y}}^T\hat{\mathbf{Y}}$. Applying the classical theorems of Mirsky and Weyl [15], we obtain bounds on the eigenvalue perturbation in terms of the Frobenius norm $\|.\|_F$ and the spectral norm $\|.\|_2$ of $\mathbf{\Delta} := \mathbf{A} - \hat{\mathbf{A}}$, respectively:

$$\epsilon_{eig} := \sqrt{\sum_{i=1}^{n} \frac{1}{n}(\hat{\lambda}_i - \lambda_i)^2} \leq \|\mathbf{\Delta}\|_F/\sqrt{n} \quad \text{and} \quad \max_i|\hat{\lambda}_i - \lambda_i| \leq \|\mathbf{\Delta}\|_2 \tag{1}$$

Applying the sin theorem and results on bounding the angle of projections to subspaces [15] (see [3] for more details), we can bound the perturbation of the residual subspace $\mathbf{C}_{ab}$ in terms of the Frobenius norm of $\mathbf{\Delta}$:

$$\|\mathbf{C}_{ab} - \hat{\mathbf{C}}_{ab}\|_F \leq \frac{\sqrt{2}\|\mathbf{\Delta}\|_F}{\nu} \tag{2}$$

where $\nu$ denotes the eigengap between the $k^{th}$ and $(k+1)^{th}$ eigenvalues of the estimated covariance matrix $\hat{\mathbf{A}}$.

To obtain practical (i.e., computable) bound on the norms of $\mathbf{\Delta}$, we derive expectation bounds instead of worst case bounds. We make the following assumptions on the error matrix $\mathbf{W}$:

1. The column vectors $\mathbf{W}_1, \ldots, \mathbf{W}_n$ are independent and radially symmetric $m$-vectors.

2. For each $i = 1, \ldots, n$, all elements of column vector $\mathbf{W}_i$ are i.i.d. random variables with mean 0, variance $\sigma_i^2 := \sigma_i^2(\delta_i)$ and fourth moment $\mu_i^4 := \mu_i^4(\delta_i)$.

Note that the independence assumption is imposed only on the error—this by no means implies that the signals received by different monitors are statistically independent. Under the above assumption, we can show that $\|\mathbf{\Delta}\|_F/\sqrt{n}$ is upper bounded in expectation by the following quantity:

$$Tol_F = 2\sqrt{\frac{1}{mn}\sum_{i=1}^{n}\lambda_i \cdot \sum_{i=1}^{n}\sigma_i^2} + \sqrt{\left(\frac{1}{m} + \frac{1}{n}\right)\sum_{i=1}^{n}\sigma_i^4 + \frac{1}{mn}\sum_{i=1}^{n}(\mu_i^4 - \sigma_i^4)}. \tag{3}$$

Similar results can be obtained for the spectral norm as well. In practice, these upper bounds are very tight because $\sigma_1, \ldots, \sigma_n$ tend to be small compared to the top eigenvalues. Given the tolerable perturbation $Tol_F$, we can use Eqn. (3) to select the slack variables. For example, we can divide the overall tolerance across monitors either uniformly or in proportion to their observed local variance.

### 3.3 Guarantee on false alarm probability

Because our approximation perturbs the eigenvalues, it also impacts the accuracy with which the trigger is fired. Since the trigger condition is $\|\mathbf{C}_{ab}\mathbf{y}\|^2 > Q_\alpha$, we must assess the impact on both of these terms. We can compute an upper bound on the perturbation of the SPE statistic, $\mathbf{SPE} = \|\mathbf{C}_{ab}\mathbf{y}\|^2$, as follows. First, note that

$$
\begin{aligned}
|\|\hat{\mathbf{C}}_{ab}\hat{\mathbf{y}}\| - \|\mathbf{C}_{ab}\mathbf{y}\|| &\leq \|(\hat{\mathbf{C}}_{ab} - \mathbf{C}_{ab})\hat{\mathbf{y}}\| + \|\mathbf{C}_{ab}(\mathbf{y} - \hat{\mathbf{y}})\| \leq \frac{\sqrt{2}\|\mathbf{\Delta}\|_F\|\hat{\mathbf{y}}\|}{\nu} + \|\mathbf{C}_{ab}\|_2\sqrt{\sum_{i=1}^{n}\delta_i^2} \\
&\leq \frac{\sqrt{2}\|\mathbf{\Delta}\|_F\|\hat{\mathbf{y}}\|}{\nu} + \left(\|\hat{\mathbf{C}}_{ab}\| + \frac{\sqrt{2}\|\mathbf{\Delta}\|_F}{\nu}\right)\sqrt{\sum_{i=1}^{n}\delta_i^2} =: \eta_1(\hat{\mathbf{y}}).
\end{aligned}
$$

$$
|\|\hat{\mathbf{C}}_{ab}\hat{\mathbf{y}}\|^2 - \|\mathbf{C}_{ab}\mathbf{y}\|^2| \leq \eta_1(\hat{\mathbf{y}})(2\|\hat{\mathbf{C}}_{ab}\hat{\mathbf{y}}\| + \eta_1(\hat{\mathbf{y}})) =: \eta_2(\hat{\mathbf{y}}). \tag{4}
$$

The dependency of the threshold $Q_\alpha$ on the eigenvalues, $\lambda_{k+1}, \ldots, \lambda_n$, can be expressed as [4]:

$$
Q_\alpha = \phi_1\left[\frac{c_\alpha\sqrt{2\phi_2 h_0^2}}{\phi_1} + 1 + \frac{\phi_2 h_0(h_0 - 1)}{\phi_1^2}\right]^{\frac{1}{h_0}}, \tag{5}
$$

where $c_\alpha$ is the $(1 - \alpha)$-percentile of the standard normal distribution, $h_0 = 1 - \frac{2\phi_1\phi_3}{3\phi_2^2}$, $\phi_i = \sum_{j=k+1}^{n}\lambda_j^i$ for $i = 1, 2, 3$.

To assess the perturbation in false alarm probability, we start by considering the following random variable $c$ derived from Eqn. (5):

$$
c = \frac{\phi_1[(\mathbf{SPE}/\phi_1)^{h_0} - 1 - \phi_2 h_0(h_0 - 1)/\phi_1^2]}{\sqrt{2\phi_2 h_0^2}}. \tag{6}
$$

The random variable $c$ essentially normalizes the random quantity $\|\mathbf{C}_{ab}\mathbf{y}\|^2$ and is known to approximately follow a standard normal distribution [5]. The false alarm probability in the centralized system is expressed as

$$
\mathbf{Pr}\left[\|\mathbf{C}_{ab}\mathbf{y}\|^2 > Q_\alpha\right] = \mathbf{Pr}\left[c > c_\alpha\right] = \alpha,
$$

where the lefthand term of this equation is conditioned upon the SPE statistics being inside the normal range. In our distributed setting, the anomaly detector fires a trigger if $\|\hat{\mathbf{C}}_{ab}\hat{\mathbf{y}}\|^2 > \hat{Q}_\alpha$. We thus only observe a perturbed version $\hat{c}$ for the random variable $c$. Let $\eta_c$ denote the bound on $|\hat{c} - c|$. The deviation of the false alarm probability in our approximate detection scheme can then be approximated as $P(c_\alpha - \eta_c < U < c_\alpha + \eta_c)$, where $U$ is a standard normal random variable.

## 4 Evaluation

We implemented our algorithm and developed a trace-driven simulator to validate our methods. We used a one-week trace collected from the Abilene network[1]. The traces contains per-link traffic loads measured every 10 minutes, for all 41 links of the Abilene network. With a time unit of 10 minutes, data was collected for 1008 time units. This data was used to feed the simulator. There are 7 anomalies in the data that were detected by the centralized algorithm (and verified by hand to be true anomalies). We also injected 70 synthetic anomalies into this dataset using the method described in [8], so that we would have sufficient data to compute error rates. We used a threshold $Q_\alpha$ corresponding to an $1 - \alpha = 99.5\%$ confidence level. Due to space limitations, we present results only for the case of uniform monitor slack, $\delta_i = \delta$.

The input parameter for our algorithm is the tolerable relative error of the eigenvalues ("relative eigen-error" for short), which acts as a tuning knob. (Precisely, it is $Tol_F/\sqrt{\frac{1}{n}\sum\lambda_i^2}$, where $Tol_F$ is defined in Eqn. (3).) Given this parameter and the input data we can compute the filtering slack $\delta$ for the monitors using Eqn. (3). We then feed in the data to run our protocol in the simulator with the

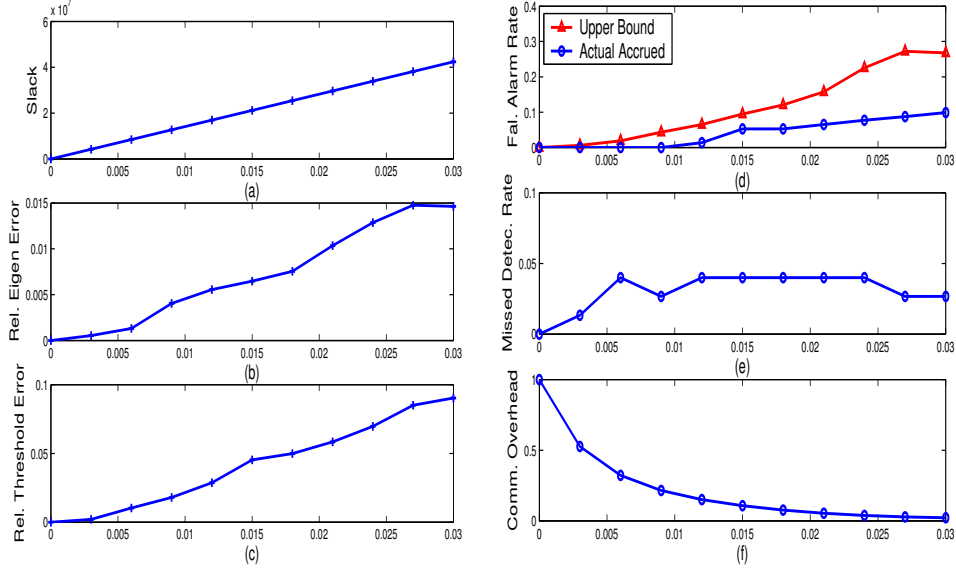

Figure 3: In all plots the $x$-axis is the relative eigen-error. (a) The filtering slack. (b) Actual accrued eigen-error. (c) Relative error of detection threshold. (d) False alarm rates. (e) Missed detection rates. (f) Communication overhead.

computed $\delta$. The simulator outputs a set of results including: 1) the actual relative eigen errors and the relative errors on the detection threshold $Q_\alpha$; 2) the missed detection rate, false alarm rate and communication cost achieved by our method. The *missed-detection rate* is defined as the fraction of missed detections over the total number of real anomalies, and the *false-alarm rate* as the fraction of false alarms over the total number of detected anomalies by our protocol, which is $\alpha$ (defined in Sec. 3.3) rescaled as a rate rather than a probability. The communication cost is computed as the fraction of number of messages that actually get through the filtering window to the coordinator.
The results are shown in Fig. 3. In all plots, the $x$-axis is the relative eigen-error. In Fig. 3(a) we plot the relationship between the relative eigen-error and the filtering slack $\delta$ when assuming filtering errors are uniformly distributed on interval $[-\delta, \delta]$. With this model, the relationship between the relative eigen-error and the slack is determined by a simplified version of Eqn. (3) (with all $\sigma_i^2 = \frac{\delta^2}{3}$). The results make intuitive sense. As we increase our error tolerance, we can filter more at the monitor and send less to the coordinator. The slack increases almost linearly with the relative eigen-error because the first term in the right hand side of Eqn. (3) dominates all other terms.

In Fig. 3(b) we compare the relative eigen-error to the actual accrued relative eigen-error (defined as $\epsilon_{eig}/\sqrt{\frac{1}{n}\sum \lambda_i^2}$, where $\epsilon_{eig}$ is defined in Eqn (1)). These were computed using the slack parameters $\delta$ as computed by our coordinator. We can see that the real accrued eigen-errors are always less than the tolerable eigen errors. The plot shows a tight upper bound, indicating that it is safe to use our model's derived filtering slack $\delta$. In other words, the achieved eigen-error always remains below the requested tolerable error specified as input, and the slack chosen given the tolerable error is close to being optimal. Fig. 3(c) shows the relationship between the relative eigen-error and the relative error of detection threshold $Q_\alpha$[2]. We see that the threshold for detecting anomalies decreases as we tolerate more and more eigen-errors. In these experiments, an error of 2% in the eigenvalues leads to an error of approximately 6% in our estimate of the appropriate cutoff threshold.

We now examine the false alarm rates achieved. In Fig. 3(d) the curve with triangles represents the upper bound on the false alarm rate as estimated by the coordinator. The curve with circles is the actual accrued false alarm rate achieved by our scheme. Note that the upper bound on the false alarm rate is fairly close to the true values, especially when the slack is small. The false alarm rate increases with increasing eigen-error because as the eigen-error increases, the corresponding detection threshold $Q_\alpha$ will decrease, which in turn causes the protocol to raise an alarm more often. (If we had plotted $\hat{Q}$ rather than the relative threshold difference, we would obviously see a

decreasing $\hat{Q}$ with increasing eigen-error.) We see in Fig. 3(e) that the missed detection rates remain below 4% for various levels of communication overhead.

The communication overhead is depicted in Fig. 3(f). Clearly, the larger the errors we can tolerate, the more overhead can be reduced. Considering these last three plots (d,e,f) together, we observe several tradeoffs. For example, when the relative eigen-error is $1.5\%$, our algorithm reduces the data sent through the network by more than 90%. This gain is achieved at the cost of approximately a 4% missed detection rate and a 6% false alarm rate. This is a large reduction in communication for a small increase in detection error. These initial results illustrate that our in-network solution can dramatically lower the communication overhead while still achieving high detection accuracy.

## 5   Conclusion

We have presented a new algorithmic framework for network anomaly detection that combines distributed tracking with PCA analysis to detect anomalies with far less data than previous methods. The distributed tracking consists of local filters, installed at each monitoring site, whose parameters are selected based upon global criteria. The idea is to track the local monitoring data only enough so as to enable accurate detection. The local filtering reduces the amount of data transmitted through the network but also means that anomaly detection must be done with limited or partial views of the global state. Using methods from stochastic matrix perturbation theory, we provided an analysis for the tradeoff between the detection accuracy and the data communication overhead. We were able to control the amount of data overhead using the the relative eigen-error as a tuning knob. To the best of our knowledge, this is the first result in the literature that provides upper bounds on the false alarm rate of network anomaly detection.

## Footnotes

[1]Abilene is an Internet2 high-performance backbone network that interconnects a large number of universities as well as a few other research institutes.

[2]Precisely, it is $1 - \hat{Q}_\alpha/Q_\alpha$, where $\hat{Q}_\alpha$ is computed from $\hat{\lambda}_{k+1}, \ldots, \hat{\lambda}_n$.

## References

[1] BAI, Z.-J., CHAN, R. AND LUK, F. Principal component analysis for distributed data sets with updating. In *Proceedings of International workshop on Advanced Parallel Processing Technologies (APPT)*, 2005.

[2] DREGER, H., FELDMANN, A., PAXSON, V. AND SOMMER, R. Operational experiences with high-volume network intrusion detection. In *Proceedings of ACM Conference on Computer and Communications Security (CCS)*, 2004.

[3] HUANG, L., NGUYEN, X., GAROFALAKIS, M., JORDAN, M., JOSEPH, A. AND TAFT, N. In-network PCA and anomaly detection. Technical Report No. UCB/EECS-2007-10, EECS Department, UC Berkeley.

[4] JACKSON, J. E. AND MUDHOLKAR, G. S. Control procedures for residuals associated with principal component analysis. In *Technometrics*, 21(3):341-349, 1979.

[5] JENSEN, D. R. AND SOLOMON, H. A Gaussian approximation for the distribution of definite quadratic forms. In *Journal of the American Statistical Association*, 67(340):898-902, 1972.

[6] KERALAPURA, R., CORMODE, G. AND RAMAMIRTHAM, J. Communication-efficient distributed monitoring of thresholded counts. In *Proceedings of ACM International Conference on Management of Data (SIGMOD)*, 2006.

[7] KREIDL, P. O., WILLSKY, A. Inference with minimal communication: A decision-theoretic variational approach. In *Proceedings of Neural Information Processing Systems (NIPS)*, 2006.

[8] LAKHINA, A., CROVELLA, M. AND DIOT, C. Diagnosing network-wide traffic anomalies. In *Proceedings of ACM Conference of the Special Interest Group on Data Communication (SIGCOMM)*, 2004.

[9] LAKHINA, A., PAPAGIANNAKI, K., CROVELLA, M., DIOT, C., KOLACZYK, E. D. AND TAFT, N. Structural analysis of network traffic flows. In *Proceedings of International Conference on Measurement and Modeling of Computer Systems (SIGMETRICS)*, 2004.

[10] LEVCHENKO, K., PATURI, R. AND VARGHESE, G. On the difficulty of scalably detecting network attacks. In *Proceedings of ACM Conference on Computer and Communications Security (CCS)*, 2004.

[11] NGUYEN, X., WAINWRIGHT, M. AND JORDAN, M. Nonparametric decentralized detection using kernel methods. In *IEEE Transactions on Signal Processing*, 53(11):4053-4066, 2005.

[12] PADMANABHAN, V. N., RAMABHADRAN, S., AND PADHYE, J. Netprofiler: Profiling wide-area networks using peer cooperation. In *Proceedings of International Workshop on Peer-to-Peer Systems*, 2005.

[13] PREDD, J.B., KULKARNI, S.B., AND POOR, H.V. Distributed learning in wireless sensor networks. In *IEEE Signal Processing Magazine*, 23(4):56-69, 2006.

[14] QU, Y., OSTROUCHOVZ, G., SAMATOVAZ, N AND GEIST, A. Principal component analysis for dimension reduction in massive distributed data sets. In *Proceedings of IEEE International Conference on Data Mining (ICDM)*, 2002.

[15] STEWART, G. W., AND SUN, J.-G. *Matrix Perturbation Theory*. Academic Press, 1990.

[16] YEGNESWARAN, V., BARFORD, P., AND JHA, S. Global intrusion detection in the domino overlay system. In *Proceedings of Network and Distributed System Security Symposium (NDSS)*, 2004.

[17] ZHANG, Y., GE, Z.-H., GREENBERG, A., AND ROUGHAN, M. Network anomography. In *Proceedings of Internet Measurement Conference (IMC)*, 2005.
